# An Information-Theoretic Approach to Deciphering the Hippocampal Code

William E. Skaggs    Bruce L. McNaughton    Katalin M. Gothard

Etan J. Markus
Center for Neural Systems, Memory, and Aging
344 Life Sciences North
University of Arizona
Tucson AZ 85724
*bill@nsma.arizona.edu*

## Abstract

Information theory is used to derive a simple formula for the amount of information conveyed by the firing rate of a neuron about any experimentally measured variable or combination of variables (e.g. running speed, head direction, location of the animal, etc.). The derivation treats the cell as a communication channel whose input is the measured variable and whose output is the cell's spike train. Applying the formula, we find systematic differences in the information content of hippocampal "place cells" in different experimental conditions.

## 1  INTRODUCTION

Almost any neuron will respond to some manipulation or other by changing its firing rate, and this change in firing can convey information to downstream neurons. The aim of this article is to introduce a very simple formula for the average rate at which a cell conveys information in this way, and to show how the formula is helpful in the study of the firing properties of cells in the rat hippocampus. This is by no means the first application of information theory to the study of neural coding; see especially Richmond and Optican (1990). The thing that particularly distinguishes

our approach is its simplemindedness.

To get the basic idea, imagine we are recording the activity of a neuron in the brain of a rat, while the rat is wandering around randomly on a circular platform. Suppose we observe that the cell fires only when the rat is on the left half of the platform, and that it fires at a constant rate everywhere on the left half; and suppose that on the whole the rat spends half of its time on the left half of the platform. In this case, if we are prevented from seeing where the rat is, but are informed that the neuron has just this very moment fired a spike, we obtain thereby one bit of information about the current location of the rat. Suppose we have a second cell, which fires only in the southwest quarter of the platform; in this case a spike would give us two bits of information. If there were in addition a small amount of background firing, the information would be slightly less than two bits. And so on.

Going back to the cell that fires everywhere on the left half of the platform, suppose that when it is active, it fires at a mean rate of 10 spikes per second. Since it is active half the time, it fires at an overall mean rate of 5 spikes per second. Since a spike conveys one bit of information about the rat's location, the cell's spike train conveys information at an average rate of 5 bits per second. This does not mean that if the cell is observed for one second, on average 5 bits will be obtained—rather it means that if the cell is observed for a sufficiently short time interval $\Delta t$, on average $5\Delta t$ bits will be obtained. In 20 milliseconds, for example, the expected information conveyed by the cell about the location of the rat will be very nearly 0.1 bits. The longer the time interval over which the cell is observed, the more redundancy in the spike train, and hence the farther below $5\Delta t$ the total information falls.

The formula that leads to these numbers is

$$I = \int_x \lambda(x) \log_2 \frac{\lambda(x)}{\lambda} \ p(x)dx, \qquad (1)$$

where $I$ is the information rate of the cell in bits per second, $x$ is spatial location, $p(x)$ is the probability density for the rat being at location $x$, $\lambda(x)$ is the mean firing rate when the rat is at location $x$, and $\lambda = \int_x \lambda(x)p(x)dx$ is the overall mean firing rate of the cell. The derivation of this formula appears in the final section. (To our knowledge the formula, though very simple, has not previously been published.)

Note that, as far as the formula is concerned, there is nothing special about spatial location: the formula can equally well be used to define the rate at which a cell conveys information about *any* aspect of the rat's state, or any combination of aspects. The only mathematical requirement[1] is that the rat's state $x$ and the spike train of the cell both be stationary random variables, so that the probability density $p(x)$ and the expected firing rate $\lambda(x)$ are well-defined.

The information rate given by formula (1) is measured in bits per second. If it is divided by the overall mean firing rate $\lambda$ of the cell (expressed in spikes per second), then a different kind of information rate is obtained, in units of bits per spike—let us call it the *information per spike*. This is a measure of the *specificity* of the cell: the more "grandmotherish" the cell, the more information per spike. For a population

of cells, then, a highly distributed representation equates to little information per spike.

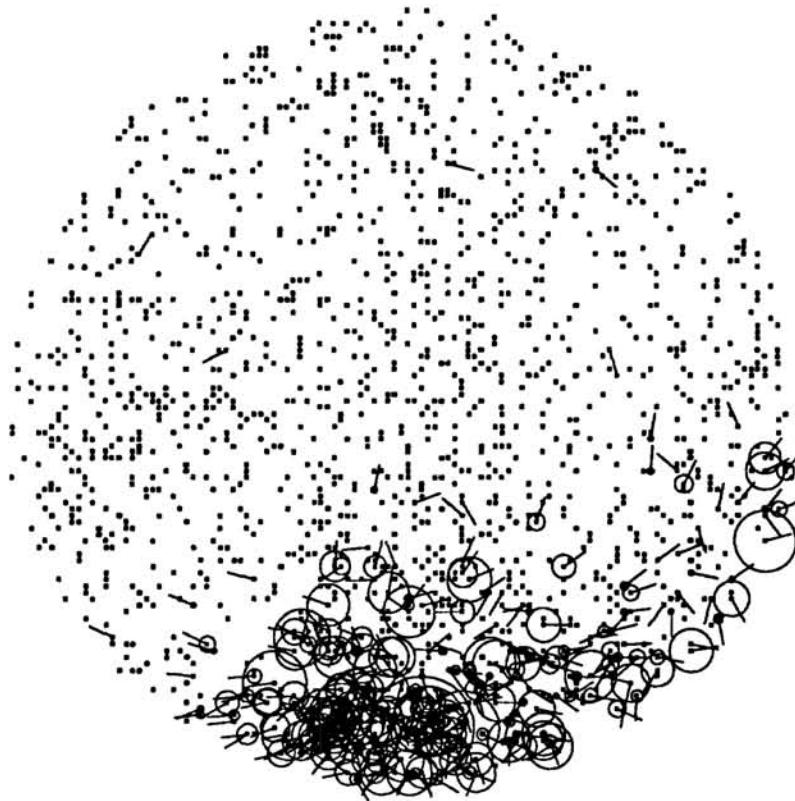

Figure 1: "Spot plot" of the activity of a single pyramidal cell in the hippocampus of a rat, recorded while the rat foraged for food pellets inside a small cylinder. The dots show locations visited by the rat, and the circles show points where the cell fired—large circles mean that several spikes occurred within a short time. The lines indicate which direction the rat was facing when the cell fired. The plot represents 29 minutes of data, during which the cell fired at an overall mean rate of 1.319 Hz.

Consider, as an example, a typical "place cell" (actually an especially nice place cell) from the CA1 layer of the hippocampus of a rat—Figure 1 shows a "spot plot" of the activity of the cell as the rat moves around inside a 76 cm diameter cylinder with high, opaque walls, foraging for randomly scattered food pellets. This cell, like most pyramidal cells in CA1, fires at a relatively high rate (above 10 Hz) when the rat is in a specific small portion of the environment—the "place field" of the cell— but at a much lower rate elsewhere. Different cells have place fields in different locations; there are no systematic rules for their arrangement, except that there may be a tendency for neighboring cells to have nearby place fields. The activity of place cells is known to be related to more than just place: in some circumstances it is sensitive to the direction the rat is facing, and it can also be modulated by running speed, alertness, or other aspects of behavioral state. The dependence on

head direction has given rise to a certain amount of controversy, because in some types of environment it is very strong, while in others it is virtually absent.

Table 1 gives statistics for the amount of information conveyed by this cell about spatial location, head direction, running speed, and combinations of these variables. Note that the information conveyed about spatial location *and* head direction is hardly more than the information conveyed about spatial location alone—the difference is well within the error bounds of the calculation. Thus this cell has no detectable directionality. This seems to be typical of cells recorded in unstructured environments.

Table 1: Information conveyed by the cell whose activity is plotted in Figure 1.

| VARIABLES | INFO | INFO PER SPIKE |
|---|---|---|
| Location | 2.40 bits/sec | 1.82 bits |
| Head Direction | 0.48 bits/sec | 0.37 bits |
| Running Speed | 0.03 bits/sec | 0.02 bits |
| Location *and* Head Direction | 2.53 bits/sec | 1.92 bits |
| Location *and* Running Speed | 2.36 bits/sec | 1.79 bits |

The information-rate measure may be helpful in understanding the computations performed by neural populations. Consider an example. Cells in the CA3 and CA1 regions of the rat hippocampal formation have long been known to convey information about a rat's spatial location (this is discussed in more detail below). Data from our lab suggest that, in a given environment, an average CA3 cell conveys something in the neighborhood of 0.1 bits per second about the rat's position—some cells convey a good deal more information than this, but many are virtually silent. Cells in CA1 receive most of their input from cells in CA3; each gets on the order of 10,000 such inputs. Question: How long must the integration time of a CA1 cell be in order for it to form a good estimate of the rat's location? Answer: With 10,000 inputs, each conveying on average 0.1 bits per second, the cell receives information at a rate of 1000 bits per second, or 1 bit per millisecond, so in 5–10 msec the cell receives enough information to form a moderately precise estimate of location.

## 2   APPLICATIONS

We now very briefly describe two experimental studies that have found differences in the spatial information content of rat hippocampal activity under different conditions. The methods used for recording the cells are described in detail in McNaughton *et al* (1989)—to summarize, the cells were recorded with stereotrodes, which are twisted pairs of electrodes, separated by about 15 microns at the tips, that pick up the extracellular electric fields generated when cells fire. A single stereotrode can detect the activity of as many as six or seven distinct hippocampal cells; spikes from different cells can be separated on the basis of their amplitudes on the two electrodes, as well as other differences in wave shape. The location of the rat was tracked using arrays of LEDs attached to their heads and a video camera on the ceiling. Spatial firing rate maps for each cell were constructed using an adaptive binning technique designed to minimize error (Skaggs and McNaughton, *submitted*),

and information rates were calculated using these firing rate maps. As a control, the spike train was randomly time-shifted relative to the sequence of locations; this was done 100 times, and the cell was deemed to have significant spatial dependence if its information rate was more than 2.29 standard deviations above the mean of the 100 control information rates.

## 2.1    EXPERIMENT: PROXIMAL VERSUS DISTAL VISUAL CUES

In this study (a preliminary account of which appears in Gothard *et al* (1992)), the activity of place cells was recorded successively in two environments, the first a 76 cm diameter cylinder with four patterned cue-cards on the high, opaque gray wall, the second a cylinder of the same shape, but with a low, transparent plexiglass wall and four patterned cue-cards on the distant black walls of the recording room. The two environments thus had the same shape, and from any given point were visually quite similar; the difference is that in one all of the visual cues were proximal to the rat, while in the other many of them were distal.

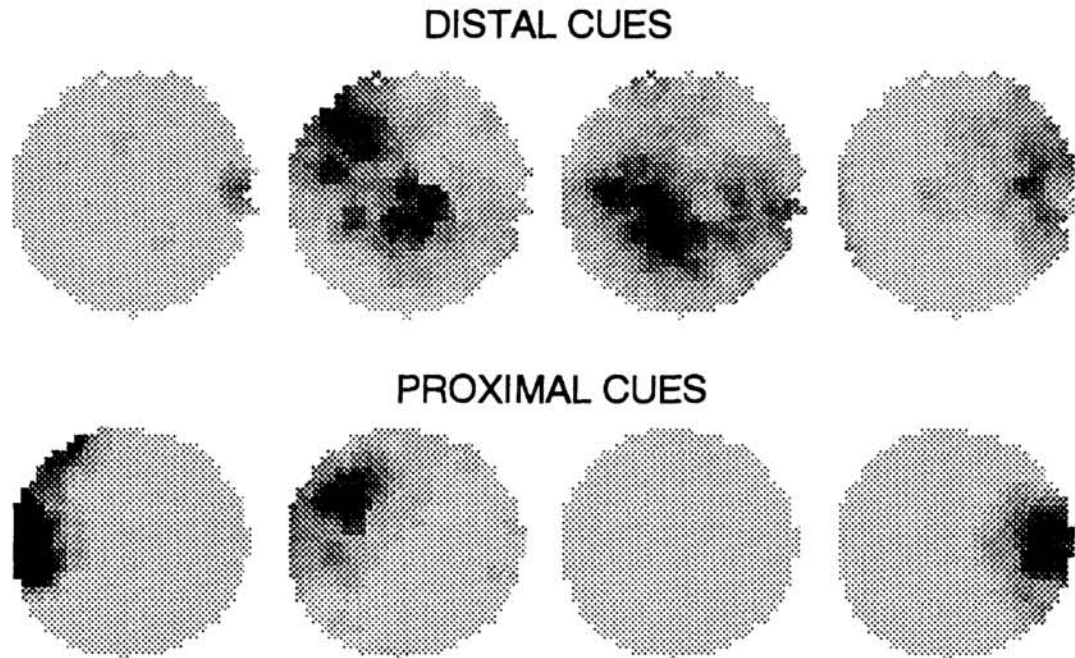

Figure 2: Firing rate maps of four simultaneously recorded cells, in the distal cue environment (top) and proximal cue environment (bottom). The scale is identical for all plots; black $\geq$ 5 Hz.

Fifty cells were recorded with robust place-dependent firing in one or the other cylinder. There was no discernable relationship between place fields in the two environments—a cell having a place field in the proximal cue environment might be nearly silent in the distal cue environment, and even if it did fire, its place field would be in a different location. (Figure 2 shows firing rate maps for four of the cells.) A substantially higher fraction of the cells had place fields in the proximal cue environment, and overall the average information per second was almost 50% higher

in the proximal cue environment. For the cells possessing fields, the information per spike was significantly higher in the proximal cue environment, meaning that place fields were more compact.

These results indicate that in the proximal cue environment, spatial location is represented by the hippocampal population more precisely, and by a larger pool of cells, than in the distal cue environment. The most likely explanation is that, at least in the absence of local cues, the configuration of visual landmarks controls the activity of the place cell population.

## 2.2  EXPERIMENT: LIGHT VERSUS DARK

Visual cues have a great deal of influence on place fields, but they are not the only important factor; in fact, some hippocampal cells maintain place fields even in complete darkness (McNaughton *et al.*, 1989b; Quirk *et al.*, 1990). This experiment (Markus *et al.*, 1992) was designed to examine how lack of visual cues changes the properties of place fields. Rats traversed an eight-arm radial maze for chocolate milk reward, with the room lights being turned on and off on alternate trials. (A trial consisted of one visit to each of the eight arms of the maze.) Figure 3 shows firing rate maps for four simultaneously recorded cells.

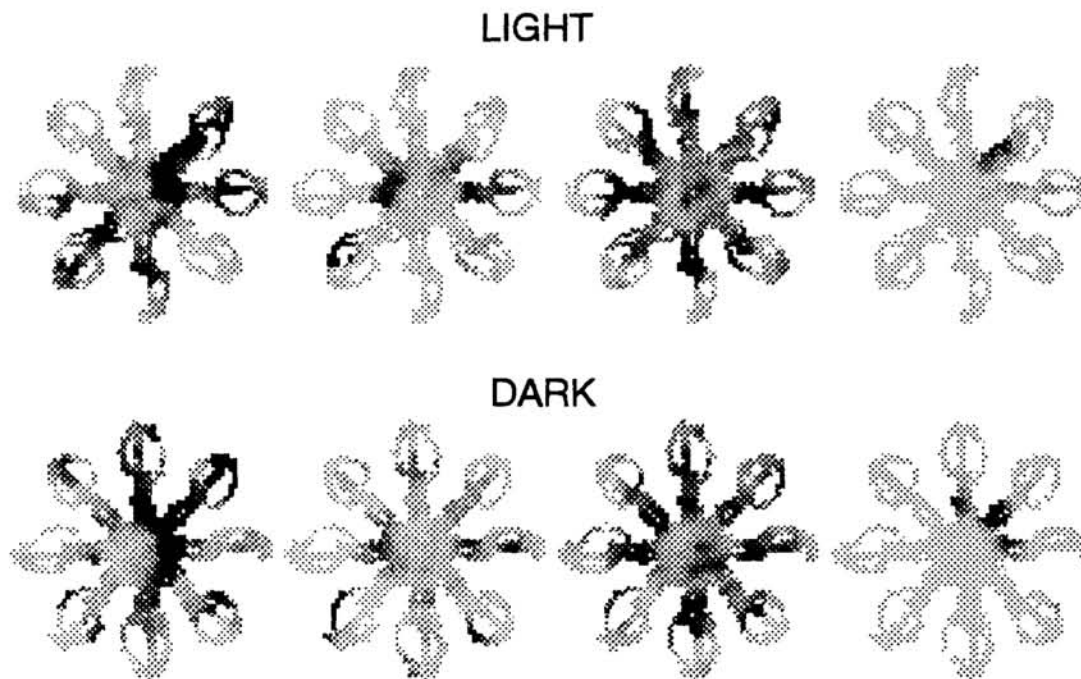

Figure 3: Firing rate maps of four simultaneously recorded cells, with room lights turned on (top) and off (bottom). The scale is identical for all plots; black $\geq$ 5 Hz. (The loops at the ends of the arms are caused by the rat turning around there.)

The most salient effect was that a much larger fraction of cells showed significant spatially selective firing in the light than in the dark: 35% as opposed to 20%. However, the average information per second decreased only by 15% in the dark as compared to the light, from 0.326 bits per second in the light to 0.278 bits per

second in the dark. (These are overestimates of the population averages, because cells silent in both light and dark were not included in the sample.)

Interestingly, the drop in information content from light to dark seemed to be much smaller than the drop from proximal cues to distal cues in the previous experiment. A major difference between the two experiments is that, in the eight-arm maze, tactile cues potentially give a great deal of information about spatial location, but in a cylinder they serve only to distinguish the center from the wall. While it is dangerous to compare the two experiments, which differed methodologically in several ways, the results suggest that tactile cues can have a very strong influence on hippocampal firing, at least when visual cues are absent.

## 3   THEORY

The information-rate formula (1) is derived by considering a neuron as a "channel" (in the information-theoretic sense) whose input is the spatial location of the rat, and whose output is the spike train. During a sufficiently short time interval the spike train is effectively a binary random variable (i.e. the only possibilities are to spike once or not at all), and the probability of spiking is determined by the spatial location. The event of spiking may be indicated by a random variable $S$ whose value is 1 if the cell spikes and 0 otherwise. If the environment is partitioned into a set of nonoverlapping bins, then spatial location may be represented by an integer-valued random variable $X$ giving the index of the currently occupied bin.

In information theory, the information conveyed by a discrete random variable $X$ about another discrete random variable $Y$, which is identical to the mutual information of $X$ and $Y$, is given by

$$I(Y|X) = \sum_{i,j} p(y_i|x_j) \log_2 \frac{p(y_i|x_j)}{p(y_i)} p(x_j),$$

where $x_j$ and $y_i$ are the possible values of $X$ and $Y$, and $p()$ is probability.

If $\lambda_j$ is the mean firing rate when the rat is in bin $j$, then the probability of a spike during a brief time interval $\Delta t$ is

$$P(S=1|X=j) = \lambda_j \Delta t.$$

Also, the overall probability of a spike is

$$P(S=1) = \lambda \Delta t,$$

where

$$\lambda = \sum_j \lambda_j p_j,$$

with $p_j = P(X=j)$.

After these expressions are plugged in to the equation for $I(Y|X)$ above, it is a matter of straightforward algebra, using power series expansions of logarithms and keeping only lower order terms, to derive a discrete approximation of equation (1).

## 4   DISCUSSION

In many situations, neurons must decide whether to fire on the basis of relatively brief samples of input, often 100 milliseconds or less. A cell cannot get much information from a single input in such a short time, so to achieve precision it needs to integrate many inputs. Formula (1) provides a measure of how much information a single input conveys about a given variable in such a brief time interval.

The formula can be applied to any type of cell that uses firing rate to convey information. The only requirement is to have enough data to get good, stable estimates of firing rates. In practice, for a hippocampal cell having a mean firing rate of around 0.5 Hz in an environment, twenty minutes of data is adequate for measuring position-dependence; and for a "theta cell" (an interneuron, firing at a considerably higher rate), very clean measurements are possible.

We have used the measure in the study of hippocampal place cells, but it might actually work better for some other types. The problem with place cells is that they fire at low overall rates, so it is time-consuming to get an adequate sample. Cortical pyramidal cells often have mean rates at least ten times faster, so it ought to be easier to get accurate numbers for them. The information measure might naturally be applied to study, for example, the changes in information content of visual cortical cells as a visual stimulus is blurred or dimmed.

**Supported by NIMH grant MH46823**

## Footnotes

[1] Other than obvious requirements of integrability that are sure to be fulfilled in natural situations.

## References

Gothard, K. M., Skaggs, W. E., McNaughton, B. L., Barnes, C. A., and Youngs, S. P. (1992). Place field specificity depends on proximity of visual cues. *Soc Neurosci Abstr*, **18**:1216. 508.10.

Markus, E. J., Barnes, C. A., McNaughton, B. L., Gladden, V., Abel, T. W., and Skaggs, W. E. (1992). Decrease in the information content of hippocampal ca1 cell spatial firing patterns in the dark. *Soc Neuroscience Abstr*, **18**:1216. 508.12.

McNaughton, B. L., Leonard, B., and Chen, L. (1989b). Cortical-hippocampal interactions and cognitive mapping: A hypothesis based on reintegration of the parietal and inferotemporal pathways for visual processing. *Psychobiology*, **17**:230–235.

McNaughton, B. L., Barnes, C. A., Meltzer, J., and Sutherland, R. J. (1989a). Hippocampal granule cells are necessary for normal spatial learning but not for spatially selective pyramidal cell discharge. *Exp Brain Res*, **76**:485–496.

Quirk, G. J., Muller, R. U., and Kubie, J. L. (1990). The firing of hippocampal place cells in the dark depends on the rat's previous experience. *J Neurosci*, **10**:2008–2017.

Richmond, B. J. and Optican, L. M. (1990). Temporal encoding of two-dimensional patterns by single units in primate primary visual cortex: Ii information transmission. *J Neurophysiol*, **64**:370–380.

